# Maximum Covariance Unfolding: Manifold Learning for Bimodal Data

**Vijay Mahadevan**
Department of ECE
University of California, San Diego
La Jolla, CA 92093
vmahadev@ucsd.edu

**Chi Wah Wong**
Department of Radiology
University of California, San Diego
La Jolla, CA 92093
cwwong@ieee.org

**Jose Costa Pereira**
Department of ECE
University of California, San Diego
La Jolla, CA 92093
josecp@ucsd.edu

**Thomas T. Liu**
Department of Radiology
University of California, San Diego
La Jolla, CA 92093
ttliu@ucsd.edu

**Nuno Vasconcelos**
Department of ECE
University of California, San Diego
La Jolla, CA 92093
nvasconcelos@ucsd.edu

**Lawrence K. Saul**
Department of CSE
University of California, San Diego
La Jolla, CA 92093
saul@cs.ucsd.edu

## Abstract

We propose maximum covariance unfolding (MCU), a manifold learning algorithm for simultaneous dimensionality reduction of data from different input modalities. Given high dimensional inputs from two different but naturally aligned sources, MCU computes a common low dimensional embedding that maximizes the cross-modal (inter-source) correlations while preserving the local (intra-source) distances. In this paper, we explore two applications of MCU. First we use MCU to analyze EEG-fMRI data, where an important goal is to visualize the fMRI voxels that are most strongly correlated with changes in EEG traces. To perform this visualization, we augment MCU with an additional step for metric learning in the high dimensional voxel space. Second, we use MCU to perform cross-modal retrieval of matched image and text samples from Wikipedia. To manage large applications of MCU, we develop a fast implementation based on ideas from spectral graph theory. These ideas transform the original problem for MCU, one of semidefinite programming, into a simpler problem in semidefinite quadratic linear programming.

## 1   Introduction

Recent advances in manifold learning and nonlinear dimensionality reduction have led to powerful, new methods for the analysis and visualization of high dimensional data [14, 1, 20, 24, 16]. These methods have roots in nonparametric statistics, spectral graph theory, convex optimization, and multidimensional scaling. Notwithstanding individual differences in motivation and approach, these methods share certain features that account for their overall popularity: (i) they generally involve few tuning parameters; (ii) they make no strong distributional assumptions; (iii) efficient algorithms exist to compute the global minima of their cost functions.

All these methods solve variants of the same basic underlying problem: given high dimensional inputs, $\{\mathbf{x_1}, \mathbf{x_2}, \ldots, \mathbf{x_n}\}$, compute low dimensional outputs $\{\mathbf{y_1}, \mathbf{y_2}, \ldots, \mathbf{y_n}\}$ that preserve certain nearness relations (e.g., local distances). Solutions to this problem have found applications in many areas of science and engineering. However, many real-world applications do not map neatly into this framework. For instance, in certain applications, aligned data is acquired from two different modalities — we refer to such data as *bimodal* — and the goal is to find low dimensional representations that capture their interdependencies.

In this paper, we investigate the use of maximum variance unfolding (MVU) [24] for the simultaneous dimensionality reduction of data from different input modalities. Though the original algorithm does not solve this problem, we show that it can be adapted to provide a compelling solution. In its original formulation, MVU computes a low dimensional embedding that maximizes the variance of its outputs, subject to constraints that preserve local distances. We explore a modification of MVU that computes a joint embedding of high dimensional inputs from different data sources. In this joint embedding, our goal is to discover a common low dimensional representation of *just those degrees of variability that are correlated across different modalities*. To achieve this goal, we design the embedding to maximize the inter-source correlation between aligned outputs while preserving the local, intra-source distances. By analogy to MVU, we call our approach *maximum covariance unfolding* (MCU).

The optimization for MCU inherits the basic form of the optimization for MVU. In particular, it can be cast as a semidefinite program (SDP). For applications to large datasets, we can also exploit the same strategies behind recent, much faster implementations of MVU [25]. In particular, using these same strategies, we show how to reformulate the optimization for MCU as a semidefinite quadratic linear program (SQLP). In addition, for one of our applications—the analysis of EEG-fMRI data— we show how to extend the basic optimization of MCU to visualize the high dimensional correlations between different input modalities. This is done by adding extra variables to the original SDP; these variables can be viewed as performing a type of metric learning in the high dimensional voxel space. In particular, they indicate which fMRI voxels (in the *high* dimensional space of fMRI images) correlate most strongly with observed changes in the EEG recordings.

As related work, we mention several other studies that have proposed SDPs to achieve different objectives than those of the original algorithm for MVU. Bowling et al [4, 5] developed a related approach known as action-respecting embedding for problems in robot localization. Song et al [18] reinterpreted the optimization criterion of MVU, then proposed an extension of the original algorithm that computes low dimensional embeddings subject to class labels or other side information. Finally, Shaw and Jebara [15, 16] have explored related SDPs to produce minimum-volume and structure-preserving embeddings; these SDPs yield much more sensible visualizations of social networks and large graphs that do not necessarily resemble a discretized manifold. Our work builds on the successes of these earlier studies and further extends the applicability of SDPs for nonlinear dimensionality reduction.

## 2 Maximum Covariance Unfolding

We propose a novel adaptation of MVU, termed maximum covariance unfolding or MCU to perform non-linear correlation between two aligned datasets whose points have a one-to-one correspondence. MCU embeds the two datasets, of different dimensions, into a single low dimensional manifold such that the two resulting embeddings are maximally correlated. As in MVU, the embeddings are such that local distances are preserved. The problem formulation is described in detail next.

### 2.1 Formulation

Let $\{\mathbf{x}_{1i}\}_{i=1}^{n}, \mathbf{x}_{1i} \in \mathcal{R}^{p_1}$ and $\{\mathbf{x}_{2i}\}_{i=1}^{n}, \mathbf{x}_{2i} \in \mathcal{R}^{p_2}$ be two aligned datasets belonging to two different input spaces, and $\{\mathbf{y}_{1i}\}_{i=1}^{n}, \mathbf{y}_{1i} \in \mathcal{R}^{d}$ and $\{\mathbf{y}_{2i}\}_{i=1}^{n}, \mathbf{y}_{2i} \in \mathcal{R}^{d}$ be the corresponding low dimensional representations (in the output space), with $d \ll p_1$ and $d \ll p_2$.

As in MVU [21], we need to find a low dimensional mapping such that the Euclidean distance between pairs of points in a local neighborhood are preserved. For each dataset $s \in \{1, 2\}$, if points $\mathbf{x}_{sj}$ and $\mathbf{x}_{sk}$ are neighbors or are common neighbors of another point, we denote an indicator variable $\eta_{s_{ij}} = 1$. The neighborhood constraints can then be written as

$$||\mathbf{y}_{si} - \mathbf{y}_{sj}||^2 = ||\mathbf{x}_{si} - \mathbf{x}_{sj}||^2 \quad \text{if } \eta_{s_{ij}} = 1 \tag{1}$$

To simplify the notation, we concatenate the output points from both datasets into one large set, $\{\mathbf{z}_i\}_{i=1}^{2n}$ containing $2n$ points, $\mathbf{z}_i = \begin{cases} \mathbf{y}_{1i} & i \le n \\ \mathbf{y}_{2(i-n)} & i > n \end{cases}$

We also define the inner-product matrix for $\{\mathbf{z}_i\}$, $K_{ij} = \mathbf{z}_i.\mathbf{z}_j$. This allows us to formulate the MCU very similarly to the MVU formulation of [21], and so we omit the details for the sake of brevity.

The distance constraint of (1) is written in the matrix form as:

$$K_{ii} - 2K_{ij} + K_{jj} \quad = \quad D_{1_{ij}}, \; \{(i,j) : i,j \le n \text{ and } \eta_{1_{ij}} = 1\} \tag{2}$$
$$K_{ii} - 2K_{ij} + K_{jj} \quad = \quad D_{2_{(i-n)(j-n)}}, \; \{(i,j) : i,j > n \text{ and } \eta_{2_{(i-n)(j-n)}} = 1\} \tag{3}$$

The centering constraint to ensure that the output points of *both datasets* are centered at the origin requires that $\sum_i \mathbf{y}_{si} = 0, \forall s \in \{1,2\}$. The equivalent matrix constraints are,

$$\sum_{ij} K_{ij} = 0, \; \forall i, j \le n \qquad \sum_{ij} K_{ij} = 0, \; \forall i, j > n \tag{4}$$

The objective function is to maximize the covariance between the low dimensional representations of the two datasets. We can use the trace of the covariance matrix as a measure of how strongly the two outputs are correlated. The average covariance can be written as:

$$\text{tr}(\text{cov}(\mathbf{y}_1, \mathbf{y}_2)) = \text{tr}(\text{E}(\mathbf{y}_1 \mathbf{y}_2^{\text{T}})) = E(\text{tr}(\mathbf{y}_1 \mathbf{y}_2^{\text{T}})) = \text{E}(\mathbf{y}_1.\mathbf{y}_2) \approx \frac{1}{n} \sum_i \mathbf{y}_{1i}.\mathbf{y}_{2i} \tag{5}$$

Combining all the constraints together with the objective function, we can write the optimization as:

**Maximize:** $\quad \sum_{ij} W_{ij} K_{ij}, \quad \text{with} \quad W = \begin{bmatrix} \mathbf{0} & \mathbf{I_n} \\ \mathbf{I_n} & \mathbf{0} \end{bmatrix}$

**subject to:** $\quad K_{ii} - 2K_{ij} + K_{jj} = D_{1_{ij}}, \; \{(i,j) : i,j \le n \text{ and } \eta_{1_{ij}} = 1\}$

$\qquad\qquad\quad K_{ii} - 2K_{ij} + K_{jj} = D_{2_{(i-n)(j-n)}}, \; \{(i,j) : i,j > n \text{ and } \eta_{2_{(i-n)(j-n)}} = 1\}$

$$K \succeq 0, \quad \sum_{ij} K_{ij} = 0, \; \forall i, j \le n, \quad \sum_{ij} K_{ij} = 0, \; \forall i, j > n \tag{6}$$

As in the original MVU formulation [21], this is a semi-definite program (SDP) and can be solved using general-purpose toolboxes such as SeDuMi [19]. The solution returned by the SDP can be used to find the coordinates in the low-dimensional embedding, $\{\mathbf{y}_{1i}\}_{i=1}^n$ and $\{\mathbf{y}_{2i}\}_{i=1}^n$, using the spectral decomposition method described in [21].

One shortcoming of the MCU formulation is that it provides no means to visualize the results. While the low-dimensional embeddings of the two datasets may be well correlated, there is no way to identify which dimensions or covariates of the data points in one modality contribute to high correlation with the points in the other modality. To address this issue, we include a novel metric learning framework in the MCU formulation, as described in the next section.

## 2.2 Metric Learning for Visualization

For each dimension in one dataset, we need to compute a measure of how much it contributes to the correlation between the datasets. This can be done using a metric learning type step applied to data of one or both modalities within the MCU formulation. In this work we describe this approach for the situation where metric learning is applied to only $\{\mathbf{x}_{1i}\}$.

The MCU formulation of Section 2 assumes that the distances between the points is Euclidean. So in the computation of nearest neighbor distances, each of the $p_1$ dimensions of $\{\mathbf{x}_{1i}\}$ receive the same weight, as shown in (1). However, inspired by the recently proposed ideas in metric learning [22], we use a more general distance metric by applying a linear transformation $T_1$ of size $p_1 \times p_1$ in the space, and then perform MCU using the transformed points, $T_1 \mathbf{x_i}$. This allows some distances to shrink/expand if that would help in increasing the correlation with $\{\mathbf{x}_{2i}\}$.

For the sake of simplicity, we choose a diagonal weight matrix $T_1$, whose diagonal entries are $\{\sigma_i\}_{i=1}^{p_1}$, $\sigma_i \ge 0, \forall i$. This allows us to weight each dimension of the input space separately.

In order to find the weight vector that produces the maximal correlation between the two datasets, these $p_1$ new variables can be learned within the MCU framework by adding them to the optimization

problem. As each dimension has a corresponding weight, the optimal weight vector returned would be a map over the dimensions indicating how strongly each is correlated to $\{\mathbf{x}_{2i}\}$.

To modify the MCU formulation to include these new variables, we replace all Euclidean distance measurements for the data points in the first dataset in (2) with the weighted distance $D_{1_{ij}} = \sum_m \sigma_m (\mathbf{x}_{im} - \mathbf{x}_{jm})^2$.

This adds a linear function of the new weight variables to the existing distance constraints of (2). However, if we had to define the neighborhood of a data point itself using this weighted distance, the formulation would become non-convex. So we assume that the neighborhood is composed of points that are closest in time . An alternative is to use neighbors as computed in the original space using the un-weighted distance. We also add constraints to make the weights positive and sum to $p_1$.

The objective function of (6) does not change, but we need to maximize the objective over the $p_1$ weight variables also. The problem still remains an SDP and can be solved as before. The new formulation, denoted MCU-ML, is written as:

**Maximize:** $\quad \sum_{ij} W_{ij} K_{ij}, \quad$ with $\quad W = \begin{bmatrix} \mathbf{0} & \mathbf{I_n} \\ \mathbf{I_n} & \mathbf{0} \end{bmatrix}$

**subject to:** $\quad \sigma_k \geq 0, \ \forall k \in \{1 \ldots p_1\}, \quad$ and $\sum_k \sigma_k = p_1.$

$$K_{ii} - 2K_{ij} + K_{jj} - \sum_m \sigma_m (\mathbf{x}_{im} - \mathbf{x}_{jm})^2 = 0, \ \{(i,j) : i,j \leq n \text{ and } \eta_{1_{ij}} = 1\}$$

$$K_{ii} - 2K_{ij} + K_{jj} = D_{2_{(i-n)(j-n)}}, \ \{(i,j) : i,j > n \text{ and } \eta_{2_{(i-n)(j-n)}} = 1\}$$

$$K \succeq 0, \quad \sum_{ij} K_{ij} = 0, \ \forall i,j \leq n, \quad \sum_{ij} K_{ij} = 0, \ \forall i,j > n \qquad (7)$$

We next describe how these formulations for MCU can be applied to find optimal representations for high dimensional EEG-fMRI data.

## 3   Resting-state EEG-fMRI Data

In the absence of an explicit task, temporal synchrony of the blood oxygenation level dependent (BOLD) signal is maintained across distinct brain regions. Taking advantage of this synchrony, resting-state fMRI has been used to study connectivity. fMRI datasets have high resolution of the order of a few millimeters, but offer poor temporal resolution as it measures the delayed haemodynamic response to neural activity. In addition, changes in resting-state BOLD connectivity measures are typically interpreted as changes in coherent neural activity across respective brain regions. However, this interpretation may be misleading because the BOLD signal is a complex function of neural activity, oxygen metabolism, cerebral blood flow (CBF), and cerebral blood volume (CBV) [3]. To address these shortcomings, simultaneous acquisition of electroencephalographic data (EEG) during functional magnetic resonance imaging (fMRI) is becoming more popular in brain imaging [13]. The EEG recording provides high temporal resolution of neural activity (5kHz), but poor spatial resolution due to electric signal distortion by the skull and scalp and the limitations on the number of electrodes that can be placed on the scalp. Therefore the goal of simultaneous acquisition of EEG and fMRI is to exploit the complementary nature of the two imaging modalities to obtain *spatiotemporally* resolved neural signal and metabolic state information [13]. Specifically, using high temporal resolution EEG data, we are able to examine dynamic changes and non-stationary properties of neural activity at different frequency bands. By correlating with the EEG data with the high resolution BOLD data, we are able to examine the corresponding spatial regions in which neural activity occurs.

Conventional approaches to analyzing the joint EEG-fMRI data have relied on *linear* methods. Most often, a simple voxel-wise correlation of the fMRI data with the EEG power time series in a specific frequency band is performed [13]. But this technique does not exploit the rich spatial dependencies of the fMRI data. To address this issue, more sophisticated linear methods such as canonical correlation analysis (CCA) [7], and the partial least squares method [11] have been proposed. However, all linear approaches have a fundamental shortcoming - the space of images, which is highly non-linear and thought to form a *manifold*, may not be well represented by a linear subspace. Therefore, lin-

ear approaches to correlate the fMRI data with the EEG data may not capture any low dimensional manifold structure.

To address these limitations we propose the use of MCU to learn low dimensional manifolds for both the fMRI and EEG data such that the output embeddings are maximally correlated. In addition, we learn a metric in the fMRI input space to identify which voxels of the fMRI correlate most strongly with observed changes in the EEG recordings. We first describe the methods used to acquire the EEG-fMRI dataset.

### 3.1 Method for Data Acquisition

One 5 minute simultaneous EEG-fMRI resting state run was recorded and processed with eyes closed (EC). Data were acquired using a 3 Tesla GE HDX system and a 64 channel EEG system supplied by Brain Products. EEG signals were recorded at 5kHz sampling rate. Impedances of the electrodes were kept below 20kΩ. Recorded EEG data were pre-processed using Vision Analyzer 2.0 software (Brain Products). Subtraction-based MR-gradient and Cardio-ballistic artifact removal were applied. A low pass filter with cut off frequency 30Hz was applied to all channels and the processed signals were down-sampled to 250Hz. fMRI data were acquired with the following parameters: echo planar imaging with 150 volumes, 30 slices, $3.438 \times 3.438 \times 5mm^3$ voxel size, $64 \times 64$ matrix size, TR=2s, TE=30ms. fMRI data were pre-processed using an in-house developed package. The 5 frequency channels of the EEG data were averaged to produce a 63 dimensional time series of 145 time points. The fMRI data consisted of a 122880 ($64 \times 64 \times 30$) dimensional time series with 145 time points.

### 3.2 Results on EEG-fMRI Dataset

The EEG and fMRI data points described in the previous section are extremely high dimensional. However, both EEG and fMRI signals are the result of sparse neuronal activity. Therefore, attempts to embed these points, especially the fMRI data, into a low dimensional manifold have been made using non-linear dimensionality reduction techniques such as Laplacian eigenmaps [17]. While such techniques may be used to find manifold embeddings for fMRI and EEG data separately, they are not useful for finding patterns of correlation between the two. We demonstrate how MCU can be applied to this setting below.

Due to the very high dimensionality of the fMRI dataset, we pre-processed the data as follows. An anatomical region of interest mask was used, followed by PCA to project the fMRI samples to a subspace of dimension $p_1 = 145$ (which represented all of the energy of the samples, because there are only 145 time points). The EEG data was not subject to any pre-processing, and $p_2$ remained 63. We applied the MCU-ML approach to learn a visualization map and a joint low dimensional embedding for the EEG-fMRI dataset. We compared the results to two other techniques - the voxel-wise correlation, and the linear CCA approach inspired by [7]. The MCU-ML solution directly returned a weight vector of length 145. For CCA, the average of the canonical directions (weighted using the canonical correlations) was used as the weight vector. In both cases, the 145 dimensional weight vector was projected back to the fMRI voxel space using the principal components of the PCA step.

Two types of voxel wise correlations maps were computed to assess the performance of MCU-ML. First, a naive correlation map was generated where each voxel was separately correlated with the average EEG power time course from the alpha aband (8-12Hz) (which is known to be correlated with the fMRI resting-state network [13]) from all the 63 electrodes. Second, a functional connectivity map was generated using the knowledge that at rest state (during which the dataset was recorded), the Posterior Cingulate Cortex (PCC) is known to be active [8] and is correlated with the Default Mode Network (DMN) while anti-correlated with the Task Positive Network (TPN). To achieve this, a seed region of interest (ROI) was first selected from PCC. The averaged fMRI signal from the ROI was then correlated with the whole brain to obtain a voxel-wise correlation map. Therefore, voxels in the PCC region should have high correlation with the EEG data. This information provides a "sanity-check" version of the fMRI correlation map.

The results for the anatomically significant slice 18, within which both DMN and the TPN are located, are shown in Figure 1. The functional connectivity map is shown in 2(a), and the correlation map obtained using MCU-ML, overlaid with the relevant anatomical regions appears in 2(b). The MCU-ML map shows the activation of Default Mode Network (DMN) and a suppression of Task Positive Network (TPN). From the results, it is clear that the MCU-ML approach produces the best

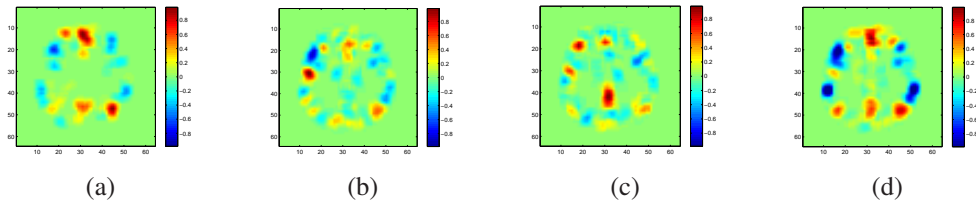

(a)          (b)          (c)          (d)

Figure 1: Comparison of results on the EEG-fMRI dataset. (a) naive correlation map (b) using only PCA (c) using CCA (d) using MCU-ML

match, showing well localized regions of positive correlation in the DMN, and regions of negative correlation in the TPN. The correlation maps for 12 slices overlaid with over a high-resolution T1-weighted image for the proposed MCU-ML approach are shown in Figure 3(b).

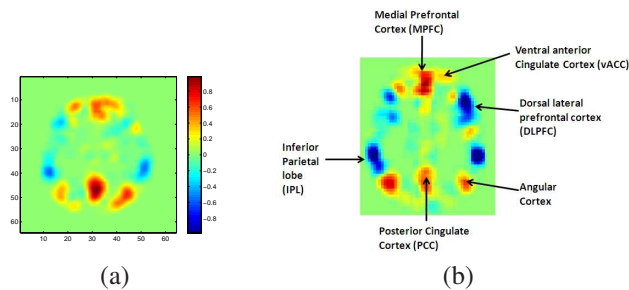

(a)          (b)

Figure 2: (a) the functional connectivity map, and (b) the MCU-ML correlation map overlaid with information about the anatomical regions relevant during rest state.

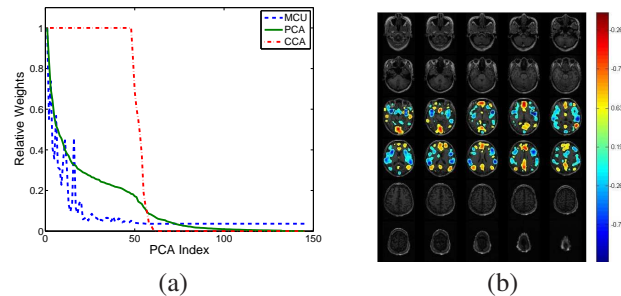

(a)          (b)

Figure 3: (a) The plot showing the normalized weights for the 145 dimensions for CCA, MCU-ML and PCA. (b) A montage showing the recovered weights for each voxel in the 12 anatomically significant slices, with the MCU overlaid on a high-resolution T1-weighted image.

To compare the learned weights using the MCU-ML and CCA, we plot the normalized importance of each of the 145 dimensions in Figure 3(a). We also plot the eigenvalues for the 145 dimensions obtained using PCA. It is seen that the weights produced by the MCU-ML approach have fewer components (around 20) than those of CCA. It is also interesting to see that the weights that produce maximal correlation with the EEG dataset are very different from the eigenvalues of PCA themselves, indicating that the dimensions that are important for correlation are not necessarily the ones with maximum variance.

## 4 Fast MCU

One of the primary limitations of the SDP based formulation for MCU in Section 2.1, shared with MVU, is its inability to scale to problems involving a large number of data points [23]. To address this issue, Weinberger et al. [23] modified the original formulation using graph laplacian regularization to reduce the size of the SDP. However, recent work has shown that even this reduced formulation of MVU can be solved more efficiently by reframing it as a semidefinite quadratic linear programming (SQLP) [25]. In this section, we show how a fast version of MCU, denoted *Fast-MCU*, can be implemented using a similar approach.

Let $L_1$ and $L_2$ denote the graph laplacians [6] of the two sets of points, $\{\mathbf{y}_{1i}\}$ and $\{\mathbf{y}_{2i}\}$, respectively. The graph laplacian depends only on nearest neighbor relations and in MCU these are assumed to be unchanged as the points are embedded from the original space to the low dimensional manifold. Therefore, $L_1$ and $L_2$ can be obtained using the graph of data points, $\{\mathbf{x}_{1i}\}$ and $\{\mathbf{x}_{2i}\}$, in the original space. Let $Q_1, Q_2 \in \mathcal{R}^{n \times m}$ contain the bottom $m$ eigenvectors of $L_1$ and $L_2$. Then we can write $2n$ vectors $\{\mathbf{y}_{1i}\}$ and $\{\mathbf{y}_{2i}\}$ in terms of two new sets of $m$ unknown vectors, $\{\mathbf{u}_1\}_{i=1}^m$ and $\{\mathbf{u}_2\}_{i=1}^m$, with $m \ll n$, using the approximation:

$$\mathbf{y}_{1i} \approx \sum_{\alpha=1}^m Q_{1_{i\alpha}} \mathbf{u}_{1\alpha} \quad \text{and} \quad \mathbf{y}_{2i} \approx \sum_{\alpha=1}^m Q_{2_{i\alpha}} \mathbf{u}_{2\alpha} \tag{8}$$

As in Section 2, we concatenate the vectors from both datasets into one larger set, $\{\mathbf{u}_i\}_{i=1}^{2m}$ containing $2m$ points:

$$\mathbf{u}_i = \begin{cases} \mathbf{u}_{1i} & i \leq m \\ \mathbf{u}_{2(i-m)} & i > m \end{cases} \tag{9}$$

We define $m \times m$ inner product matrices, $(U_{ij})_{\alpha\beta} = \mathbf{u}_{i\alpha}^T \mathbf{u}_{j\beta} \ \forall i,j \in \{1,2\} \ \forall \alpha, \beta \in \{1\ldots m\}$, and a $2m \times 2m$ matrix, $U_{\alpha\beta} = \mathbf{u}_\alpha^T \mathbf{u}_\beta \ \forall \alpha, \beta \in \{1\ldots 2m\}$. Therefore, $U = \begin{bmatrix} U_{11} & U_{12} \\ U_{21} & U_{22} \end{bmatrix}$.

The $2n \times 2n$ inner product matrix $K$ can therefore be approximated in terms of the much smaller matrix $2m \times 2m$ matrix $U$:

$$K \approx \begin{bmatrix} Q_1 U_{11} Q_1^T & Q_1 U_{21} Q_2^T \\ Q_2 U_{21} Q_1^T & Q_2 U_{22} Q_2^T \end{bmatrix} \tag{10}$$

The formulate MCU as an SQLP, we first rewrite (6) by bringing the distance constraints into the objective function using regularization parameters $\nu_1, \nu_2 > 0$:

**Maximize:** $\displaystyle\sum_{ij} W_{ij} K_{ij} - \nu_1 \sum_{i\sim j, \forall i,j \leq n} \left(K_{ii} - 2K_{ij} + K_{jj} - D_{1_{ij}}\right)^2$

$$- \nu_2 \sum_{i\sim j, \forall i,j > n} \left(K_{ii} - 2K_{ij} + K_{jj} - D_{2_{ij}}\right)^2$$

**subject to:** $\displaystyle K \succeq 0, \quad \sum_{ij} K_{ij} = 0, \ \forall i,j \leq n, \quad \sum_{ij} K_{ij} = 0, \ \forall i,j > n \tag{11}$

By using (10) in (11), and by noting that the centering constraint is automatically satisfied [23], we get the modified formulation in terms of $U$:

**Maximize:** $\displaystyle 2\mathrm{tr}(Q_1 U_{21} Q_2^T) - \sum_k \nu_k \sum_{i\sim_k j} \left((Q_k U_{kk} Q_k^T)_{ii} - 2(Q_k U_{kk} Q_k^T)_{ij} + (Q_k U_{kk} Q_k^T)_{jj} - D_{k_{ij}}\right)^2$

**subject to:** $\quad U \succeq 0 \tag{12}$

where $i \sim_k j$ for $k \in \{1,2\}$ encodes the neighborhood relationships of the $k^{th}$ dataset.

This SDP is similar to the formulation proposed by [23]. In order to obtain further simplification, let $\mathcal{U} \in \mathcal{R}^{4m^2}$ be the concatenation of the columns of $U$. Then, (12) can be reformulated by collecting the coefficients of all quadratic terms in the objective function in a positive semi-definite matrix $A \in \mathcal{R}^{4m^2 \times 4m^2}$, and those of the linear terms, including the trace term, in a vector $b \in R^{4m^2}$:

**Minimize:** $\quad \mathcal{U} A \mathcal{U}^T + b^T \mathcal{U}$

**subject to:** $\quad U \succeq 0 \tag{13}$

This minimization problem can be solved using the SQLP approach of [6]. From the solution of the SQLP, the vectors $\{\mathbf{u}_{1i}\}_{i=1}^m$ and $\{\mathbf{u}_{2i}\}_{i=1}^m$, can be obtained using the spectral decomposition method described in [21], followed by the low dimensional coordinates $\{\mathbf{y}_{1i}\}_{i=1}^n$ and $\{\mathbf{y}_{2i}\}_{i=1}^n$, using (8). Finally, these coordinates are refined using gradient based improvement of the original objective function of (11) using the procedure described in [23].

## 5  Results

We apply the Fast-MCU algorithm to $n = 1000$ points generated from two "Swiss rolls" in three dimensions, with $m$ set to 20. Figure 4 shows the embeddings of this data generated by CCA and

by Fast-MCU. While CCA discovers two significant dimensions, the Fast-MCU accurately extracts the low dimensional manifold where the embeddings lie in a narrow strip.

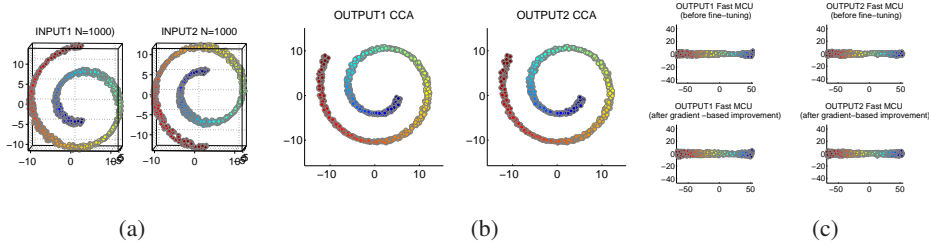

(a)                    (b)                    (c)

Figure 4: (a) Two "swiss rolls" consisting of 1000 points each in 3D with the aligned pairs of points shown in the same color. (b) the 2D embedding obtained using CCA. (c) low dimensional manifolds obtained using Fast-MCU, before and after the gradient based improvement step. (best viewed in color)

To further test the proposed Fast-MCU on real data, we use the recently proposed Wikipedia dataset composed of text and image pairs [12]. The dataset consists of 2866 text - image pairs, each belonging to one of 10 semantic categories. The corpus is split into a training set with 2173 documents, and a test set with 693 documents. The retrieval task consists of two parts. In the first, each image in the test set is used as a query, and the goal is to rank all the texts in the test set based on their match to the query image. In the second, a text query is used to rank the images. In both parts, performance is measured using the mean average precision (MAP). The MAP score is the average precision at the ranks where recall changes.

The experimental evaluation was similar to that of [12]. We first represented the text using an LDA model [2] with 20 topics, and the image using a histogram over a SIFT [10] codebook of 4096 codewords. The common low dimensional manifold was learned from the text-image pairs of the training set using the SQLP based formulation of (13), with $m = 20$, followed by a gradient ascent step as described in the previous section. To compare the performance of Fast-MCU, we also used CCA and kernel CCA (kCCA) to learn the maximally correlated joint spaces from the training set. For kCCA we used a Gaussian kernel and implemented it using code from the authors of [9].

Given a test sample (image or text), it is first projected into the learned subspace or manifold. For CCA, this involves a linear transformation to the low dimensional subspace, while for kCCA this is achieved by evaluating a linear combination of the kernel functions of the training points [9]. For Fast-MCU, the nearest neighbors of the test point among the training samples in the original space are used to obtain a mapping of the point as a weighted combination of these neighbors. The same mapping is then applied to the projection of the neighbors in the learned low dimensional joint manifold to compute the projection of the test point. To perform retrieval, all the test points of both modalities, image and text, are projected to the joint space learned using the training set. For a given test point of one modality, its distance to all the projected test points of the other modality are computed, and these are then ranked. In this work, we used the normalized correlation distance, which was shown to be the best performing distance metric in [12]. A retrieved sample is considered to be correct if it belongs to the same category as the query.

The results of the retrieval task are shown in Table 1. The performance of a random retrieval scheme is also shown to indicate the baseline chance level. It is clear that Fast-MCU outperforms both CCA and kCCA in both image-to-text and text-to-image retrieval tasks. In addition, Fast-MCU produced significantly lower number of dimensions for the embeddings - CCA produced 19 signficant dimensions compared to just 3 for Fast-MCU.

Table 1: MAP Scores for image-text retrieval tasks

| Query | Random | CCA | KCCA | Fast-MCU |
|---|---|---|---|---|
| Text - Image | 0.118 | 0.193 | 0.170 | **0.264** |
| Image - Text | 0.118 | 0.154 | 0.172 | **0.198** |

## 6 Conclusions

In this paper, we describe an adaptation of MVU to analyze correlation of high-dimensional aligned data such as EEG-fMRI data and image-text corpora. Our results on EEG-fMRI data show that

the proposed approach is able to make anatomically significant predictions about which voxels of the fMRI are most correlated with changes in EEG signals. Likewise, the results on the Wikipedia set demonstrate the ability of MCU to discover the correlations between images and text. In both these applications, it is important to realize that MCU is not only revealing the correlated degrees of variability from different input modalities, but also pruning away the uncorrelated ones. This ability of MCU makes it much more broadly applicable because in general we expect inputs from truly different modalities to have many independent degrees of freedom: e.g., there are many ways in text to describe a single, particular image, just as there are many ways in pictures to illustrate a single, particular word.

# 7 Acknowledgements

This work was supported by NSF award CCF-0830535, NIH Grant R01NS051661 and ONR MURI Award No. N00014-10-1-0072.

# References

[1] M. Belkin and P. Niyogi. Laplacian eigenmaps for dimensionality reduction and data representation. *Neural Computation*, 15(6):1373–1396, 2003.

[2] D. Blei, A. Ng, and M. Jordan. Latent dirichlet allocation. *JMLR* , 3:993–1022, 2003.

[3] R. Buxton, K. Uluda, D. Dubowitz, and T. T Liu. Modeling the hemodynamic response to brain activation. *Neuroimage*, 23(1):220-233, 2004.

[4] M. Bowling, A. Ghodsi, and D. Wilkinson. Action respecting embedding. In *ICML*, pages 65–72, 2005.

[5] M. Bowling, D. Wilkinson, A. Ghodsi, and A. Milstein. Subjective localization with action respecting embedding. In *ISRR*, 2005.

[6] F. Chung. *Spectral graph theory*. Amer Mathematical Society, 1997.

[7] N. Correa, T. Eichele, T. AdalI, Y. Li, and V. Calhoun. Multi-set canonical correlation analysis for the fusion of concurrent single trial ERP and functional MRI. *NeuroImage*, 2010.

[8] M. Greicius, B. Krasnow, A. Reiss, and V. Menon. Functional connectivity in the resting brain: a network analysis of the default mode hypothesis. *PNAS*, 100(1):253, 2003.

[9] D. Hardoon, S. Szedmak, and J. Shawe-Taylor. Canonical correlation analysis: An overview with application to learning methods. *Neural Computation*, 16(12):2639–2664, 2004.

[10] D. Lowe. Distinctive image features from scale-invariant keypoints. *IJCV*, 60(2):91–110, 2004.

[11] E. Martinez-Montes, P. Valdés-Sosa, F. Miwakeichi, R. Goldman, and M. Cohen. Concurrent EEG/fMRI analysis by multiway partial least squares. *NeuroImage*, 22(3):1023–1034, 2004.

[12] N. Rasiwasia, J. Costa Pereira, E. Coviello, G. Doyle, G. Lanckriet, R. Levy, and N. Vasconcelos. A new approach to cross-modal multimedia retrieval. In *ACM Multimedia*, pages 251–260, 2010.

[13] P. Ritter and A. Villringer. Simultaneous EEG-fMRI. *Neuroscience & Biobehavioral Reviews*, 30(6):823–838, 2006.

[14] S. T. Roweis and L. K. Saul. Nonlinear dimensionality reduction by locally linear embedding. *Science*, 290:2323–2326, 2000.

[15] B. Shaw and T. Jebara. Minimum volume embedding. In *AISTATS*, pages 460–467, San Juan, Puerto Rico, 2007.

[16] B. Shaw and T. Jebara. Structure preserving embedding. In *ICML*, 2009.

[17] X. Shen and F. Meyer. Low-dimensional embedding of fMRI datasets. *Neuroimage*, 41(3):886–902, 2008.

[18] L. Song, A. Smola, K. Borgwardt, and A. Gretton. Colored maximum variance unfolding. *NIPS* 2008.

[19] J. Sturm. Using SeDuMi 1.02, a MATLAB toolbox for optimization over symmetric cones. *Optimization methods and software*, 11(1):625–653, 1999.

[20] J. B. Tenenbaum, V. de Silva, and J. C. Langford. A global geometric framework for nonlinear dimensionality reduction. *Science*, 290:2319–2323, 2000.

[21] K. Weinberger and L. Saul. Unsupervised learning of image manifolds by semidefinite programming. *IJCV*, 70(1):77–90, 2006.

[22] K. Weinberger and L. Saul. Distance metric learning for large margin nearest neighbor classification. *JMLR*, 10:207–244, 2009.

[23] K. Weinberger, F. Sha, Q. Zhu, and L. Saul. Graph laplacian regularization for large-scale semidefinite programming. *NIPS*, 19:1489, 2007.

[24] K. Q. Weinberger, F. Sha, and L. K. Saul. Learning a kernel matrix for nonlinear dimensionality reduction. *ICML*, 2004.

[25] X. Wu, A. So, Z. Li, and S. Li. Fast graph laplacian regularized kernel learning via semidefinite–quadratic–linear programming. *NIPS*, 22:1964–1972.

